# DYNAMIC, NON-LOCAL ROLE BINDINGS AND INFERENCING IN A LOCALIST NETWORK FOR NATURAL LANGUAGE UNDERSTANDING*

Trent E. Lange
Michael G. Dyer

Artificial Intelligence Laboratory
Computer Science Department
University of California, Los Angeles
Los Angeles, CA 90024

## ABSTRACT

This paper introduces a means to handle the critical problem of non-local role-bindings in localist spreading-activation networks. Every conceptual node in the network broadcasts a stable, uniquely-identifying activation pattern, called its *signature*. A dynamic role-binding is created when a role's binding node has an activation that matches the bound concept's signature. Most importantly, signatures are propagated across long paths of nodes to handle the non-local role-bindings necessary for inferencing. Our localist network model, ROBIN (ROle Binding and Inferencing Network), uses signature activations to robustly represent schemata role-bindings and thus perform the inferencing, plan/goal analysis, schema instantiation, word-sense disambiguation, and dynamic re-interpretation portions of the natural language understanding process.

## MOTIVATION

Understanding natural language is a difficult task, often requiring a reader to make multiple inferences to understand the motives of actors and to connect actions that are unrelated on the basis of surface semantics alone. An example of this is the sentence:

S1: *"John put the pot inside the dishwasher because the police were coming."*

A complex plan/goal analysis of S1 must be made to understand the actors' actions and disambiguate *"pot"* to MARIJUANA by overriding the local context of *"dishwasher"*.

## DISTRIBUTED SPREADING-ACTIVATION NETWORKS

Distributed connectionist models, such as [McClelland and Kawamoto, 1986] and [Touretzky and Hinton, 1985], are receiving much interest because their models are closer to the neural level than symbolic systems, such as [Dyer, 1983]. Despite this attention, no distributed network has yet exhibited the ability to handle natural language input having complexity even near to that of S1. The primary reason for this current lack of success is the inability to perform dynamic role-bindings and to propagate these binding constraints during inferencing. Distributed networks, furthermore, are sequential at the knowledge level and lack the representation of structure needed to handle complex conceptual relationships [Feldman, 1986].

## LOCALIST SPREADING-ACTIVATION NETWORKS

Localist spreading-activation networks, such as [Cottrell and Small, 1983] and [Waltz and Pollack, 1985], also seem more neurally plausible than symbolic logic/Lisp-based systems. Knowledge is represented in localist networks by simple computational nodes and their interconnections, with each node standing for a distinct concept. Activation on a conceptual node represents the amount of *evidence* available for that concept in the current context.

Unlike distributed networks, localist networks are parallel at the knowledge level and are able to represent structural relationships between concepts. Because of this, many different inference paths can be pursued simultaneously; a necessity if the quick responses that people are able to generate is to be modelled.

Unfortunately, however, the evidential activation on the conceptual nodes of previous localist networks gives no clue as to *where* that evidence came from. Because of this, previous localist models have been similar to distributed connectionist models in their inability to handle dynamic, non-local bindings -- and thus remain unsuited to higher-level knowledge tasks where inferencing is required.

## ROBIN

Our research has resulted in ROBIN (ROle Binding and Inferencing Network), a localist spreading-activation model with additional structure to handle the dynamic role-bindings and inferencing needed for building in-depth representations of complex and ambiguous sentences, such as S1. ROBIN's networks are built entirely with simple computational elements that clearly have the possibility of realization at the neural level.

Figure 1 shows an overview of a semantic network embedded in ROBIN after input for sentence S1 has been presented. The network has made the inferences necessary to form a plan/goal analysis of the actors' actions, with the role-bindings being instantiated dynamically with activation. The final interpretation selected is the most highly-activated path of frames inside the darkly shaded area.

As in previous localist models, ROBIN's networks have a node for every known concept

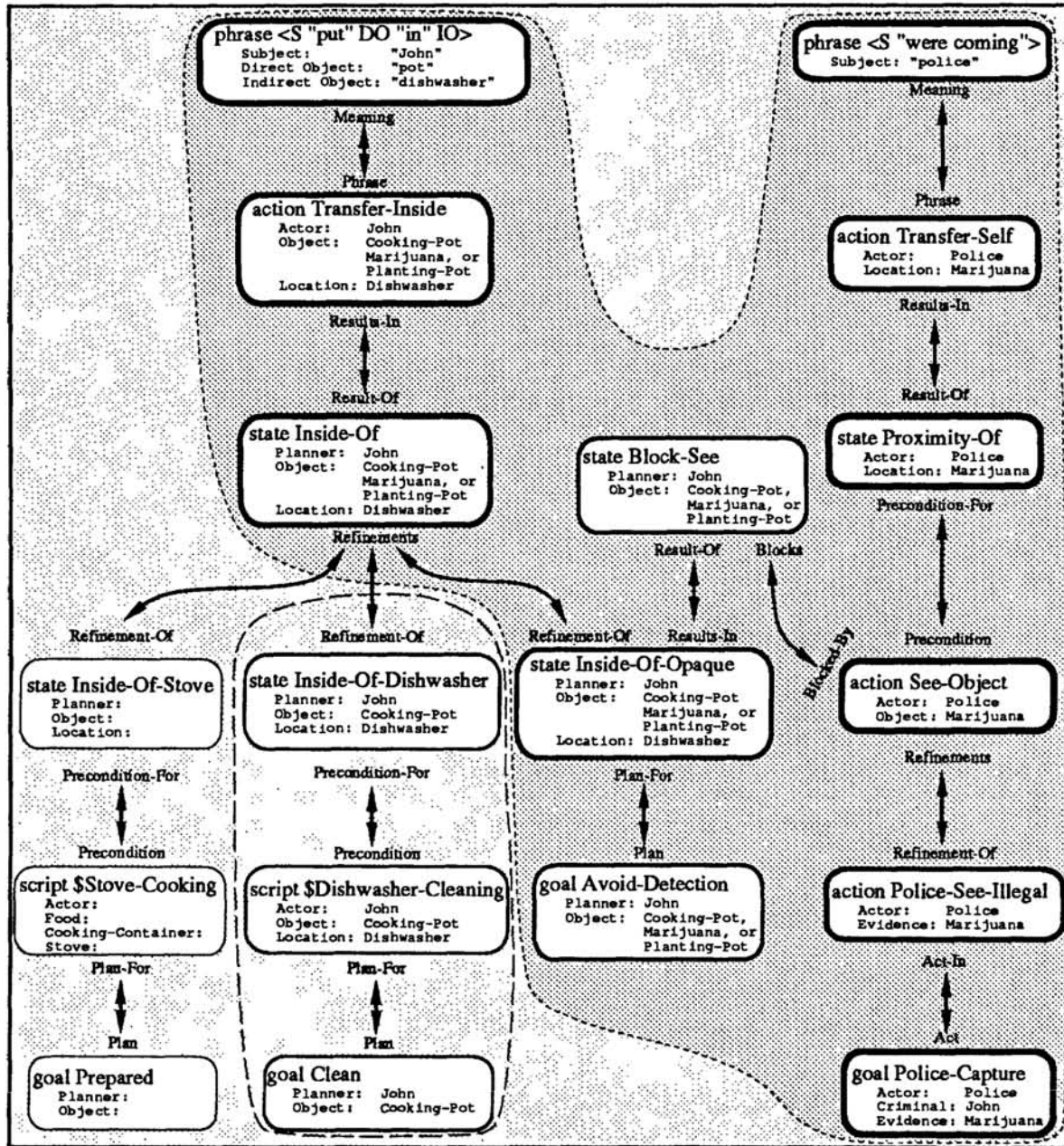

**Figure 1.** Semantic network embedded in ROBIN, showing inferences dynamically made after S1 is presented. Thickness of frame boundaries shows their amount of evidential activation. Darkly shaded area indicates the most highly-activated path of frames representing the most probable plan/goal analysis of the input. Dashed area shows discarded dishwasher-cleaning interpretation. Frames outside of both areas show a small portion of the network that received no evidential or signature activation. Each frame is actually represented by the connectivity of a set of nodes.

in the network. Relations between concepts are represented by weighted connections between their respective nodes. The activation of a conceptual node is *evidential*,

corresponding to the amount of evidence available for the concept and the likelihood that it is selected in the current context.

Simply representing the amount of evidence available for a concept, however, is not sufficient for complex language understanding tasks. Role-binding requires that some means exist for *identifying* a concept that is being dynamically bound to a role in distant areas of the network. A network may have never heard about JOHN having the goal of AVOID-DETECTION of his MARIJUANA, but it must be able to infer just such a possibility to understand S1.

## SIGNATURE ACTIVATION IN ROBIN

Every conceptual node in ROBIN's localist network has associated with it an identification node broadcasting a stable, uniquely-identifying activation pattern, called its *signature*. A dynamic binding is created when a role's *binding node* has an activation that matches the activation of the bound concept's signature node.

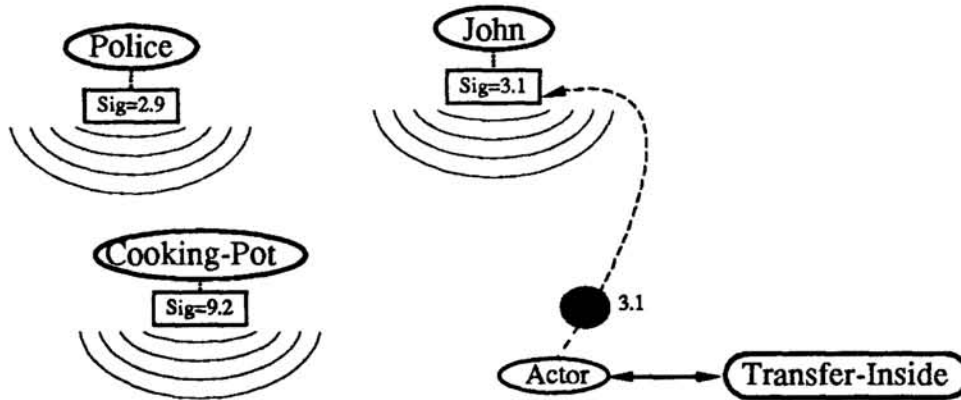

**Figure 2.** Several concepts and their uniquely-identifying signature nodes are shown, along with the Actor role of the TRANSFER-INSIDE frame. The dotted arrow from the binding node (black circle) to the signature node of JOHN represents the virtual binding indicated by the shared signature activation, and does not exist as an actual connection.

In Figure 2, the virtual binding of the Actor role node of action TRANSFER-INSIDE to JOHN is represented by the fact that its binding node, the solid black circle, has the same activation (3.1) as JOHN's signature node.

## PROPAGATION OF SIGNATURES FOR ROLE-BINDING

The most important feature of ROBIN's signature activations is that the model passes them, as activation, across long paths of nodes to handle the non-local role-bindings necessary for inferencing. Figure 3 illustrates how the structure of the network automatically accomplishes this in a ROBIN network segment that implements a portion of the semantic network of Figure 1.

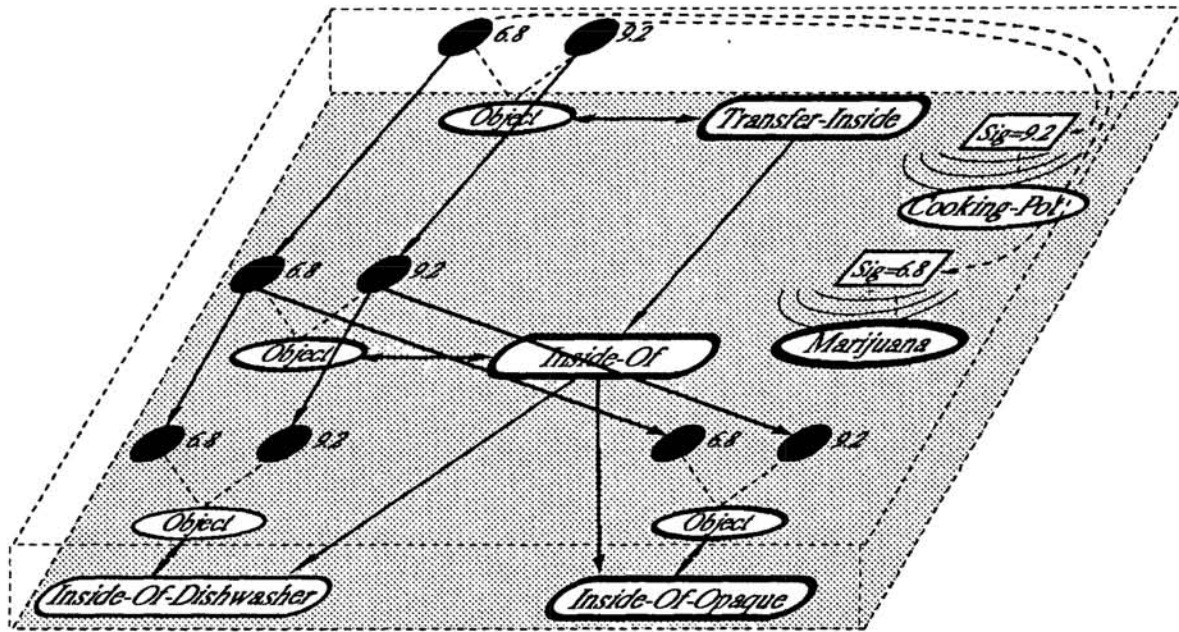

**Figure 3.** Simplified ROBIN network segment showing parallel paths over which evidential activation (bottom plane) and signature activation (top plane) are spread for inferencing. Signature nodes (rectangles) and binding nodes (solid black circles) are in the top plane. Thickness of conceptual node boundaries (ovals) represents their level of evidential activation after quiescence has been reached for sentence S1. (The names on the nodes are not used by ROBIN in any way, being used simply to set up the network's structure initially and to aid in analysis.)

Evidential activation is spread through the paths between conceptual nodes on the bottom plane (i.e. TRANSFER-INSIDE and its Object role), while signature activation for dynamic role-bindings is spread across the parallel paths of corresponding binding nodes on the top plane. Nodes and connections for the Actor, Planner, and Location roles are not shown. Initially there is no activation on any of the conceptual or binding nodes in the network.

When input for S1 is presented, the concept TRANSFER-INSIDE receives evidential activation from the phrase *"John put the pot inside the dishwasher"*, while the binding nodes of its Object role get the activations (6.8 and 9.2) of the signatures for MARIJUANA and COOKING-POT, representing the candidate bindings from the word *"pot"*.

As activation starts to spread, INSIDE-OF receives evidential activation from TRANSFER-INSIDE, representing the strong evidence that something is now inside of something else. Concurrently, the signature activations on the binding nodes of TRANSFER-INSIDE's Object propagate to the corresponding binding nodes of INSIDE-OF's Object. *The network has thus made the crucial inference of exactly which thing is inside of the other.* Similarly, as time goes on, INSIDE-OF-DISHWASHER and INSIDE-OF-OPAQUE receive evidential activation, with inferencing continuing by the propagation of signature activation to their corresponding binding nodes.

## SPREAD OF ACTIVATION IN SENTENCE S1

The rest of the semantic network needed to understand S1 (Figure 1) is also built utilizing the structure of Figure 3. Both evidential and signature activation continue to spread from the phrase *"John put the pot inside the dishwasher"*, propagating along the chain of related concepts down to the CLEAN goal, with some reaching goal AVOID-DETECTION. The phrase *"because the police were coming"* then causes evidential and signature activation to spread along a path from TRANSFER-SELF to both goals POLICE-CAPTURE and AVOID-DETECTION, until the activation of the network finally settles.

## SELECTING AMONG CANDIDATE BINDINGS

In Figure 3, signature activations for both of the ambiguous meanings of the word *"pot"* were propagated along the Object roles, with MARIJUANA and COOKING-POT being the *candidate* bindings for the role. The network's interpretation of which concept is selected at any given time is the binding whose concept has greater evidential activation. Because all candidate bindings are spread along the network, with none being discarded until processing is completed, ROBIN is easily able to handle meaning re-interpretations without resorting to backtracking. For example, a re-interpretation of the word *"pot"* back to COOKING-POT occurs when S1 is followed by *"They were coming over for dinner."*

During the interpretation of S1, COOKING-POT initially receives more evidential activation than MARIJUANA by connections from the highly stereotypical usage of the dishwasher for the CLEAN goal. The network's decision between the two candidate bindings at that point would be that it was a COOKING-POT that was INSIDE-OF the DISH-WASHER. However, reinforcement and feedback from the inference paths generated by the POLICE's TRANSFER-SELF eventually cause MARIJUANA to win out. The final selection of MARIJUANA over the COOKING-POT bindings is represented simply by the fact that MARIJUANA has greater evidential activation. The resulting most highly-activated path of nodes and non-local bindings represents the plan/goal analysis in Figure 1. A more detailed description of ROBIN's network structure can be found in [Lange, 1989].

## EVIDENTIAL VS SIGNATURE ACTIVATION

It is important to emphasize the differences between ROBIN's evidential and signature activation. Both are simply activation from a computational point of view, but they propagate across separate pathways and fulfil different functions.

Evidential Activation:

1) *Previous work* -- Similar to the activation of previous localist models.
2) *Function* -- Activation on a node represents the amount of evidence available for a node and the likelihood that its concept is selected in the current context.
3) *Node pathways* -- Spreads along weighted evidential pathways between related frames.
4) *Dynamic structure* -- Decides among candidate structures; i.e. in Figure 1, MARIJUANA is more highly-activated than COOKING-POT, so is selected as the currently most plausible role-binding throughout the inference path.

Signature Activation:

1) *Previous work* -- First introduced in ROBIN.
2) *Function* -- Activation on a node is part of a unique pattern of signature activation representing a dynamic, virtual binding of the signature's concept.
3) *Node pathways* -- Spreads along role-binding paths between corresponding roles of related frames.
4) *Dynamic structure* -- Represents a potential (candidate) dynamic structure; i.e., that either MARIJUANA or COOKING-POT is INSIDE-OF a DISHWASHER.

## NETWORK BUILDING BLOCKS AND NEURAL PLAUSIBILITY

ROBIN builds its networks with elements that each perform a simple computation on their inputs: summation, summation with thresholding and decay, multiplication, or maximization. The connections between units are either weighted excitatory or inhibitory. Max units, i.e. those outputting the maximum of their inputs, are used because of their ability to pass on signature activations without alteration.

ROBIN's most controversial element will likely be the signature-producing nodes that generate the uniquely-identifying activations upon which dynamic role-binding is based. These identifier nodes need to broadcast their unique signature activation throughout the time the concept they represent is active, and be able to broadcast the same signature whenever needed. Reference to neuroscience literature [Segundo *et al.*, 1981, 1964] reveals that self-feedbacking groups of "pacemaker" neurons have roughly this ability:

> "The mechanism described determines stable patterns in which, over a clearly defined frequency range, the output discharge is locked in phase and frequency..." [Segundo *et al.*, 1964]

Similar to pacemakers are central pattern generators (CPGs) [Ryckebusch *et al.*, 1988], which produce different stable patterns of neuronal oscillations. Groups of pacemakers or CPGs could conceivably be used to build ROBIN's signature-producing nodes, with oscillator phase-locking implementing virtual bindings of signatures. In any case, the simple computational elements ROBIN is built upon appear to be as neurally plausible as those of current distributed models.

## FUTURE WORK

There are several directions for future research: (1) *Self-organization of network structure* -- non-local bindings allow ROBIN to create novel network instances over its pre-existing structure. Over time, repeated instantiations should cause modification of weights and recruitment of underutilized nodes to alter the network structure. (2) *Signature dynamics* -- currently, the identifying signatures are single arbitrary activations; instead, signatures should be distributed patterns of activation that are learned adaptively over time, with similar concepts possessing similar signature patterns.

# CONCLUSION

This paper describes ROBIN, a domain-independent localist spreading-activation network model that approaches many of the problems of natural language understanding, including those of inferencing and frame selection. To allow this, the activation on the network's simple computational nodes is of one of two types: (a) *evidential activation*, to indicate the likelihood that a concept is selected, and (b) *signature activation*, to uniquely identify concepts and allow the representation and propagation of dynamic virtual role-bindings not possible in previous localist or distributed models.

ROBIN's localist networks use the spread of evidential and signature activation along their built-in structure of simple computational nodes to form a single most highly-activated path representing a plan/goal analysis of the input. It thus performs the inferencing, plan/goal analysis, schema instantiation, word-sense disambiguation, and dynamic re-interpretation tasks required for natural language understanding.

## Footnotes

*This research is supported in part by a contract with the JTF program of the DOD and grants from the ITA Foundation and the Hughes Artificial Intelligence Center.

## References

Cottrell, G. & Small, S. (1982): A Connectionist Scheme for Modeling Word-Sense Disambiguation. *Cognition and Brain Theory, 6*, p. 89-120.

Dyer, M. G. (1983): *In-Depth Understanding: A Computer Model of Integrated Processing for Narrative Comprehension*, MIT Press, Cambridge, MA.

Feldman, J. A. (1986): *Neural Representation of Conceptual Knowledge*, (Technical Report TR189), Department of Computer Science, University of Rochester.

Lange, T. (1989): (forthcoming) *High-Level Inferencing in a Localist Network*, Master's Thesis, Department of Computer Science, University of California, Los Angeles.

McClelland, J. L. & Kawamoto, A. H. (1986): Mechanisms of Sentence Processing: Assigning Roles to Constituents of Sentences. In McClelland & Rumelhart (eds.) *Parallel Distributed Processing: Vol 2*. Cambridge, MA: The MIT Press.

Ryckebusch, S., Mead, C., & Bower, J. M. (1988): Modeling a Central Pattern Generator in Software and Hardware: Tritonia in Sea Moss. *Proceedings of IEEE Conference on Neural Information Processing Systems -- Natural and Synthetic (NIPS-88)*, Denver, CO.

Segundo, J. P., Perkel, D. H., Schulman, J. H., Bullock, T. H., & Moore, G. P (1964): Pacemaker Neurons: Effects of Regularly Spaced Synaptic Input. *Science, Volume 145, Number 3627*, p. 61-63.

Segundo, J. P. & Kohn, A. F. (1981): A Model of Excitatory Synaptic Interactions Between Pacemakers. Its Reality, its Generality, and the Principles Involved. *Biological Cybernetics, Volume 40*, p. 113-126.

Touretzky, D. S. & Hinton, G. E. (1985): Symbols among the Neurons: Details of a Connectionist Inference Architecture. *Proceedings of the International Joint Conference on Artificial Intelligence*, Los Angeles, CA.

Waltz, D. & Pollack, J. (1985): Massively Parallel Parsing: A Strongly Interactive Model of Natural Language Interpretation. *Cognitive Science, Volume 9, Number 1*, p. 51-74.